# Range Image Restoration using Mean Field Annealing

Griff L. Bilbro                    Wesley E. Snyder

Center for Communications and Signal Processing
North Carolina State University
Raleigh, NC

## Abstract

A new optimization strategy, Mean Field Annealing, is presented. Its application to MAP restoration of noisy range images is derived and experimentally verified.

# 1  Introduction

The application which motivates this paper is image analysis; specifically the analysis of range images. We [BS86] [GS87] and others [YA85][BJ88] have found that surface curvature has the potential for providing an excellent, view-invariant feature with which to segment range images. Unfortunately, computation of curvature requires, in turn, computation of second derivatives of noisy data.

We cast this task as a restoration problem: Given a measurement $g(x, y)$, we assume that $g(x, y)$ resulted from the addition of noise to some "ideal" image $f(x, y)$ which we must estimate from three things:

1. The measurement $g(x, y)$.
2. The statistics of the noise, here assumed to be zero mean with variance $\sigma^2$.
3. Some *a priori* knowledge of the smoothness of the underlying surface(s).

We will turn this restoration problem into a minimization, and solve that minimization using a strategy called *Mean Field Annealing*. A neural net appears to be the ideal architecture for the resulting algorithm, and some work in this area has already been reported [CZVJ88].

# 2  Simulated Annealing and Mean Field Annealing

The strategy of SSA may be summarized as follows:
Let $H(f)$ be the objective function whose minimum we seek, where $f$ is some parameter vector.
A parameter $T$ controls the algorithm. The SSA algorithm begins at a relatively high value of $T$ which is gradually reduced. Under certain conditions, SSA will converge to a global optimum, [GG84] [RS87]

$$H(f) = min\{H(f^k)\} \forall f^k \tag{1}$$

even though local minima may occur. However, SSA suffers from two drawbacks:

- It is slow, and
- there is no way to directly estimate [MMP87] a continuously-valued $f$ or its derivatives.

The algorithm presented in section 2.1 perturbs (typically) a single element of $f$ at each iteration. In Mean Field Annealing, we perturb the entire vector $f$ at each iteration by making a deterministic calculation which lowers a certain average of $H$, $< H(f) >$, *at the current temperature*. We thus perform a rather conventional non-linear minimization (e.g. gradient descent), until a minimum is found at that temperature. We will refer to the minimization condition at a given $T$ as the *equilibrium* for that $T$. Then, $T$ is reduced, and the previous equilibrium is used as the initial condition for another minimization.

MFA thus converts a hard optimization problem into a sequence of easier problems. In the next section, we justify this approach by relating it to SSA.

## 2.1    Stochastic Simulated Annealing

The problem to be solved is to find $\hat{f}$ where $\hat{f}$ minimizes $H(f)$. SSA solves this minimization with the following strategy:

1. Define $p_T \propto e^{-H/T}$.
2. Find the equilibrium conditions on $p_T$, at the current temperature, T. By equilibrium, we mean that any statistic of $p_T(f)$ is constant. These statistics could be derived from the Markov chain which SSA constructs: $f^0, f^1, ..., f^N, ...$, although in fact such statistical analysis is never done in normal running of an SSA algorithm.
3. Reduce $T$ gradually.
4. As $T \to 0$, $p_T(f)$ becomes sharply peaked at $\hat{f}$, the minimum.

## 2.2    Mean Field Annealing

In Mean Field Annealing, we provide an analytic mechanism for approximating the equilibrium at arbitrary $T$. In MFA, we define an error function,

$$E_{MF}(x, T) = Tln \frac{\int e^{\frac{-H}{T}} df}{\int e^{\frac{-H_0}{T}} df} + \frac{\int e^{\frac{-H_0}{T}} (H - H_0) df}{\int e^{-\frac{H_0}{T}} df}. \tag{2}$$

which follows from Peierl's inequality [BGZ76]:

$$F \le F_0 + < H - H_0 > \tag{3}$$

where $F = -Tln \int e^{\frac{-H}{T}} df$ and $F_0 = -Tln \int e^{\frac{-H_0}{T}} df$. The significance of $E_{MF}$ is as follows: the minimum of $E_{MF}$ determines the best approximation given the form

of $H_0$ to the equilibrium statistics of the SSA-generated MRF at T. We will then anneal on T. In the next section, we choose a special form for $H_0$ to simplify this process even further.

1. Define some $H_0(f, x)$ which will be used to estimate $H(f)$.
2. At temperature $T$, minimize $E_{MF}(x)$ where $E_{MF}$ is a functional of $H_0$ and $H$ which characterizes the difference between $H_0$ and $H$. The process of minimizing $E_{MF}$ will result in a value of the parameter $x$, which we will denote as $\hat{x}_T$.
3. Define $\hat{H}_T(f) = H_0(f, \hat{x}_T)$ and $\hat{p}_T(f) \propto e^{-\hat{H}_T/T}$.

## 3    Image Restoration Using MFA

We choose a Hamiltonian which represents both the noise in the image, and our *a priori* knowledge of the local shape of the image data.

$$H_N = \sum_i \frac{1}{2\sigma^2}(f_i - g_i)^2 \qquad (4)$$

$$H_P = \sum_i V(\Lambda(f_{\partial i})) \qquad (5)$$

where $f_{\partial i}$ represents [Bes86] the set of values of pixels neighboring pixel $i$ (e.g. the value of $f$ at $i$ along with the $f$ values at the four nearest neighbors of $i$); $\Lambda$ is some scalar valued function of that set of pixels (e.g. the 5 pixel approximation to the Laplacian or the 9 pixel approximation to the quadratic variation); and

$$V(\Lambda) = \frac{-b}{\sqrt{2\pi\tau}} e^{\frac{-\Lambda^2}{2\tau}}. \qquad (6)$$

The noise term simply says that the image should be similar to the data, given noise of variance $\sigma^2$. The prior term drives toward solutions which are locally planar. Recently, a simpler $V(x) = x^2$ and a similar $\Lambda$ were successfully used to design a neural net [CZVJ88] which restores images consisting of discrete, but 256-valued pixels. Our formulation of the prior term emphasizes the importance of "point processes," as defined [WP85] by Wolberg and Pavlidis. While we accept the eventual necessity of incorporating line processes [MMP87] [Mar85] [GG84] [Gem87] into restoration, our emphasis in this paper is to provide a rigorous relationship between a point process, the prior model, and the more usual mathematical properties of surfaces. Using range imagery in this problem makes these relationships direct. By adopting this philosophy, we can exploit the results of Grimson [Gri83] as well as those of Brady and Horn [BH83] to improve on the Laplacian.

The Gaussian functional form of $V$ is chosen because it is mathematically convenient for Boltzmann statistics and because it reflects the following shape properties recommended for grey level images in the literature and is especially important if

line processes are to be omitted: Besag [Bes86] notes that "to encourage smooth variation", $V(\Lambda)$ "should be strictly increasing" in the absolute value of its argument and if "occasional abrupt changes" are expected, it should "quickly reach a maximum".

Rational functions with shapes similar to our $V$ have been used in recent stochastic approaches to image processing [GM85]. In Eq. 6, $\tau$ is a "soft threshold" which represents our prior knowledge of the probability of various values of $\nabla^2 f$ (the Laplacian of the undegraded image). For $\tau$ large, we imply that high values of the Laplacian are common – $f$ is highly textured; for small values of $\tau$, we imply that $f$ is generally smooth. We note that for high values of $\tau$, the prior term is insignificant, and the best estimate of the image is simply the data.

We choose the *Mean Field Hamiltonian* to be

$$H_0 = \sum_i \frac{1}{2}(f_i - x_i)^2, \tag{7}$$

and find that the optimal $\hat{x}_T$ approximately minimizes

$$H(x) \approx H(<f>) = \sum_i \frac{(x_i - g_i)^2}{2\sigma^2} - \frac{b}{\sqrt{2\pi(\tau+T)}} \sum_i e^{-\frac{(\sum_\nu L_\nu x_{\nu+i})^2}{2(\tau+T)}}. \tag{8}$$

both at very high and very low T. We have found experimentally that this approximation to $\hat{x}_T$ does anneal to a satisfactory restoration. At each temperature, we use gradient descent to find $\hat{x}_T$ with the following approximation to the gradient of $< H >$:

$$r_i = \sum_\nu L_\nu x_{i+\nu} \tag{9}$$

and

$$V(r_i) = \frac{-b}{\sqrt{2\pi(\tau+T)}} e^{-\frac{r_i^2}{2(\tau+T)}}. \tag{10}$$

Differentiating Eq. 8 with this new notation, we find

$$\frac{\partial < H >}{\partial x_j} = \frac{x_j - g_j}{\sigma^2} + \sum_i V'(r_i)(\sum_\nu L_\nu \delta_{i+\nu,j}). \tag{11}$$

Since $\delta_{i+\nu,j}$ is non-zero only when $i + \nu = j$, we have

$$\frac{\partial < H >}{\partial x_j} = \frac{x_j - g_j}{\sigma^2} + \sum_\nu L_{-\nu} V'(r_{j+\nu}) \tag{12}$$

and this derivative can be used to find the equilibrium condition.

## Algorithm

1. Initially, we use the high temperature assumption, which eliminates the prior term entirely, and results in

$$x_j = g_j; \text{ for } T = \infty. \tag{13}$$

This will provide the initial estimate of $x$. Any other estimate quickly converges to $g$.

2. Given an image $x_j$, form the image $r_j = (L \otimes x)_j$, where the $\otimes$ indicates convolution.

3. Create the image $V_p = V'(r_j) = -\frac{b}{\sqrt{2\pi(\tau+T)}} \frac{r_j}{T+\tau} e^{-\frac{r_j^2}{2(T+\tau)}}$.

4. Using 12, perform ordinary non-linear minimization of $< H >$ starting from the current $x$. The particular strategy followed is not critical. We have successfully used steepest descent and more sophisticated conjugate gradient [PFTV88] methods. The simpler methods seem adequate for Gaussian noise.

5. Update $x$ to the minimizing $\hat{x}$ found in step 4.

6. Reduce $T$ and go to 2. When $T$ is sufficiently close to 0, the algorithm is complete.

In step 6 above, $\tau$ essentially defines the appropriate low-temperature stopping point. In section 5, we will elaborate on the determination of $\tau$ and other such constants.

# 4 Performance

In this section, we describe the performance of the algorithm as it is applied to several range images. We will use range images, in which the data is of the form

$$z = z(x, y). \tag{14}$$

## 4.1 Images With High Levels of Noise

Figure 1 illustrates a range image consisting of three objects, a wedge (upper left), a cylinder with rounded end and hole (right), and a trapezoidal block viewed from the top. The noise in this region is measured at $\sigma = 3$units out of a total range of about 100 units. Unsophisticated smoothing will not estimate second derivatives of such data without blurring. Following the surface interpolation literature, [Gri83] [BH83] we use the quadratic variation as the argument of the penalty function for the prior term to

$$r^2 = (\frac{\partial^2 f}{\partial x^2})^2 + (\frac{\partial^2 f}{\partial y^2})^2 + 2(\frac{\partial^2 f}{\partial x \partial y})^2 \tag{15}$$

and performing the derivative in a manner analogous to Eq. 11 and 12. The Laplacian of the restoration is shown in Figure 2. Figure 3 shows a cross-section taken as indicated by the red line on Figure 2.

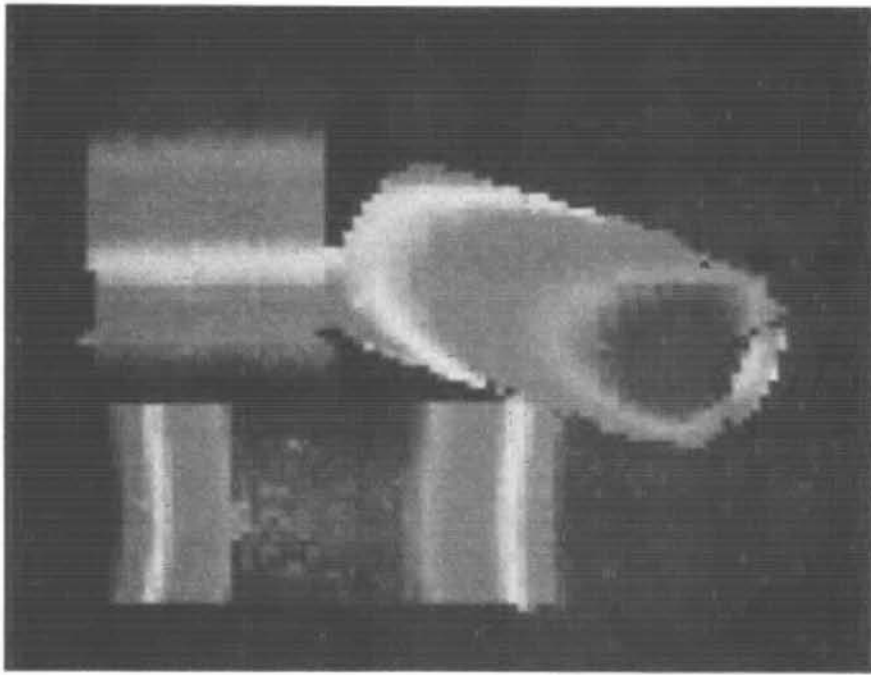

Fig. 1 Original range image

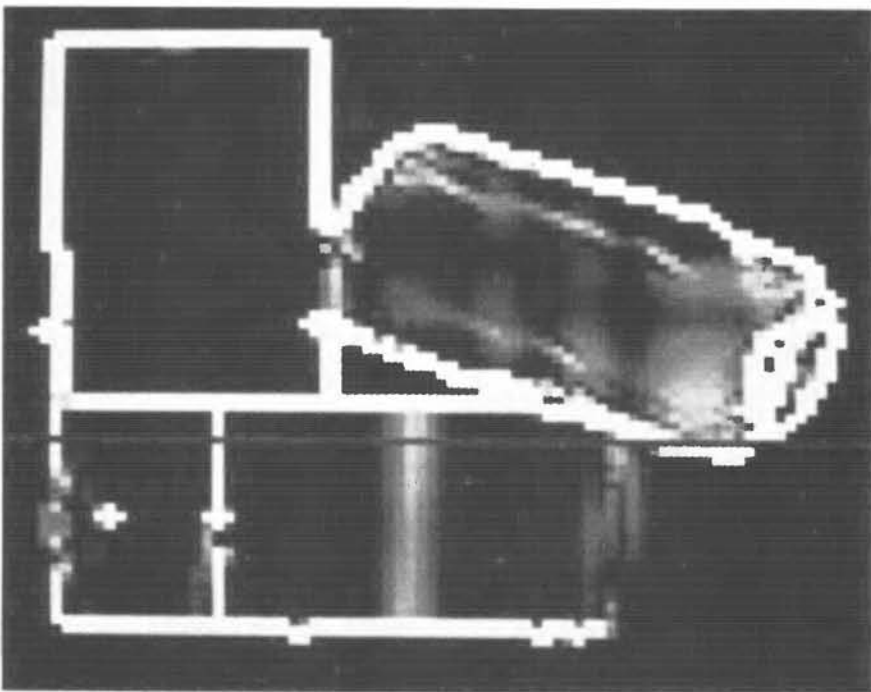

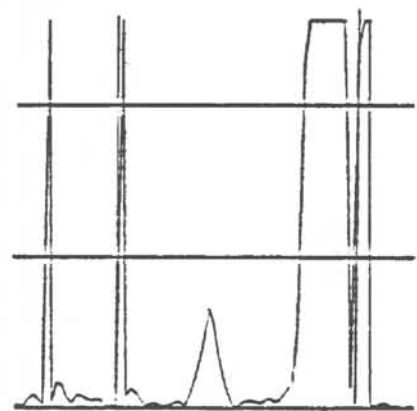

Fig. 3 Cross section
Through Laplacian along
Red Line

Fig. 2 Laplacian of the restored image

## 4.2  Comparison With Existing Techniques

Accurate computation of surface derivatives requires extremely good smoothing of
surface noise, while segmentation requires preservation of edges. One such adaptive
smoothing technique,[Ter87] iterative Gaussian smoothing (IGS) has been success-
fully applied to range imagery. [PB87] Following this strategy, step edges are first
detected, and smoothing is then applied using a small center-weighted kernel. At
edges, an even smaller kernel, called a "molecule", is used to smooth right up to
the edge without blurring the edge. The smoothing is then iterated.

The results, restoration and Laplacian, of IGS are not nearly as sharp as those shown in Figure 2.

# 5   Determining the Parameters

Although the optimization strategy described in section 3 has no hard thresholds, several parameters exist either explicitly in Eq. 8 or implicitly in the iteration. Good estimates of these parameters will result in improved performance, faster convergence, or both. The parameters are:

$\sigma$ the standard deviation of the noise
$b$ the relative magnitude of the prior term
$\tau_I = \tau + T$ the initial temperature and
$\tau$ the final temperature.

The decrement in $T$ which defines the annealing schedule could also be considered a parameter. However, we have observed that 10% or less per step is always good enough.

We find that for depth images of polyhedral scenes, $\tau = 0$ so that only one parameter is problem dependent: $\sigma$. For the more realistic case of images which also contain curved surfaces, however, see our technical report [BS88], which also describes the MFA derivation in much more detail.

The standard deviation of the noise must be determined independently for each problem class. It is straightforward to estimate $\sigma$ to within 50%, and we have observed experimentally that performance of the algorithm is not sensitive to this order of error.

We can analytically show that annealing occurs in the region $T \approx |L|^2\sigma^2$ and choose $\tau_I = 2|L|^2\sigma^2$. Here, $|L|^2$ is the squared norm of the operator $L$ and $|L|^2 = 20$ for the usual Laplacian and $|L|^2 = 12.5$ for the quadratic variation.

Further analysis shows that $b = \sqrt{2\pi}|L|\sigma$ is a good choice for the coefficient of the prior term.

# References

[Bes86]  J. Besag. On the statistical analysis of dirty pictures. *Journal of the Royal Statistical Society*, B 48(3), 1986.

[BGZ76]  E. Brezin, J. C. Le Guillon, and J. Zinn-Justin. Field theoretical approach to critical phenomena. In C. Domb and M.S. Green, editors, *Phase Transitions and Critical Phenomena*, chapter 3, Academic Press, New York, 1976.

[BH83]  M. Brady and B.K.P Horn. Symmetric operators for surface interpolation. *CVGIP*, 22, 1983.

[BJ88]  P.J. Besl and R.C. Jain. Segmentation through variable-order surface fitting. *IEEE PAMI*, 10(2), 1988.

[BS86]    G. Bilbro and W. Snyder. *A Linear Time Theory for Recognizing Surfaces in 3-D*. Technical Report CCSP-NCSU TR-86/8, Center for Communications and Signal Processing, North Carolina State University, 1986.

[BS88]    G. L. Bilbro and W. E. Snyder. *Range Image Restoration Using Mean Field Annealing*. Technical Report NETR 88-19, Available from the Center for Communications and Signal Processing, North Carolina State University, 1988.

[CZVJ88]  R. Chellappa, Y.-T. Zhou, A. Vaid, and B. K. Jenkins. Image restoration using a neural network. *IEEE Transactions on ASSP*, 36(7):1141–1151, July 1988.

[Gem87]   D. Geman. Stochastic model for boundary detection. *Vision and Image Computing*, 5(2):61–65, 1987.

[GG84]    D. Geman and S. Geman. Stochastic relaxation, Gibbs Distributions, and the Bayesian restoration of images. *IEEE Transactions on PAMI*, PAMI-6(6):721–741, November 1984.

[GM85]    S. Geman and D.E. McClure. Bayesian image analysis: an application to single photon emission tomography. *Proceedings of the American Statistical Association, Statistical Computing Section*, 12–18, 1985.

[Gri83]   W.E.L. Grimson. An implementation of computational theory of visual surface interpolation. *CVGIP*, 22, 1983.

[GS87]    B. R. Groshong and W. E. Snyder. Range image segmentation for object recognition. In *18th Pittsburgh Conference on Modeling and Simulation*, Pittsburgh, PA, April 1987.

[Mar85]   J.L. Marroquin. *Probabilistic Solution to Inverse Problems*. PhD thesis, M.I.T, Cambridge, MA, September 1985.

[MMP87]   J. Marroquin, S. Mitter, and T. Poggio. Probabilistic solution of ill-posed problems in computational vision. *Journal of American Statistical Association*, 82(397):76–89, March 1987.

[PB87]    T. Ponce and M. Brady. Toward a surface primal sketch. In T. Kanade, editor, *Three Dimensional Machine Vision*, Kluwer Press, 1987.

[PFTV88]  W. H. Press, B. P. Flannery, S. A. Teukolsky, and W. T. Vetterling. *Numerical Recipes in C*. Cambridge University Press, 1988.

[RS87]    F. Romeo and A. Sangiovanni-Vencentelli. Probabalistic hill climbing algorithms: properties and applications. In *Chapel Hill Conference on VLSI*, Computer Science Press, Chapel Hill, NC, 1987.

[Ter87]   D. Terzopoulos. The role of constraints and discontiuities in visible-surface reconstruction. In *Proc. of 7th International Conf. on AI*, pages 1073–1077, 1987.

[WP85]    G. Wolberg and T. Pavlidis. Restoration of binary images using stochastic relaxation with annealing. *Pattern Recognition Letters*, 3(6):375–388, December 1985.

[YA85]    M. Brady A. Yiulle and H. Asada. Describing surfaces. *CVGIP*, August 1985.